# The Infinite Hidden Markov Model

**Matthew J. Beal**    **Zoubin Ghahramani**    **Carl Edward Rasmussen**

Gatsby Computational Neuroscience Unit
University College London
17 Queen Square, London WC1N 3AR, England
http://www.gatsby.ucl.ac.uk
{m.beal,zoubin,edward}@gatsby.ucl.ac.uk

## Abstract

We show that it is possible to extend hidden Markov models to have
a countably infinite number of hidden states. By using the theory of
Dirichlet processes we can implicitly integrate out the infinitely many
transition parameters, leaving only three hyperparameters which can be
learned from data. These three hyperparameters define a hierarchical
Dirichlet process capable of capturing a rich set of transition dynamics.
The three hyperparameters control the time scale of the dynamics, the
sparsity of the underlying state-transition matrix, and the expected num-
ber of distinct hidden states in a finite sequence. In this framework it
is also natural to allow the alphabet of emitted symbols to be infinite—
consider, for example, symbols being possible words appearing in En-
glish text.

## 1   Introduction

Hidden Markov models (HMMs) are one of the most popular methods in machine
learning and statistics for modelling sequences such as speech and proteins. An
HMM defines a probability distribution over sequences of observations (symbols) $\mathbf{y} = \{y_1, \ldots, y_t, \ldots, y_T\}$ by invoking another sequence of unobserved, or *hidden*, discrete
state variables $\mathbf{s} = \{s_1, \ldots, s_t, \ldots, s_T\}$. The basic idea in an HMM is that the se-
quence of hidden states has Markov dynamics—i.e. given $s_t$, $s_\tau$ is independent of $s_\rho$
for all $\tau < t < \rho$—and that the observations $y_t$ are independent of all other variables
given $s_t$. The model is defined in terms of two sets of parameters, the transition matrix
whose $ij^{\text{th}}$ element is $P(s_{t+1} = j | s_t = i)$ and the emission matrix whose $iq^{\text{th}}$ element
is $P(y_t = q | s_t = i)$. The usual procedure for estimating the parameters of an HMM is
the Baum-Welch algorithm, a special case of EM, which estimates expected values of two
matrices $n$ and $m$ corresponding to counts of transitions and emissions respectively, where
the expectation is taken over the posterior probability of hidden state sequences [6].

Both the standard estimation procedure and the model definition for HMMs suffer from
important limitations. First, maximum likelihood estimation procedures do not consider
the complexity of the model, making it hard to avoid over or underfitting. Second, the
model structure has to be specified in advance. Motivated in part by these problems there
have been attempts to approximate a full Bayesian analysis of HMMs which integrates over,
rather than optimises, the parameters. It has been proposed to approximate such Bayesian
integration both using variational methods [3] and by conditioning on a single most likely
hidden state sequence [8].

In this paper we start from the point of view that the basic modelling assumption of HMMs—that the data was generated by some discrete state variable which can take on one of several values—is unreasonable for most real-world problems. Instead we formulate the idea of HMMs with a countably infinite number of hidden states. In principle, such models have infinitely many parameters in the state transition matrix. Obviously it would not be sensible to optimise these parameters; instead we use the theory of Dirichlet processes (DPs) [2, 1] to *implicitly integrate them out*, leaving just three hyperparameters defining the prior over transition dynamics.

The idea of using DPs to define mixture models with infinite number of components has been previously explored in [5] and [7]. This simple form of the DP turns out to be inadequate for HMMs.[1] Because of this we have extended the notion of a DP to a two-stage hierarchical process which couples transitions between different states. It should be stressed that Dirichlet *distributions* have been used extensively both as priors for mixing proportions and to smooth n-gram models over finite alphabets [4], which differs considerably from the model presented here. To our knowledge no one has studied inference in discrete infinite-state HMMs.

We begin with a review of Dirichlet processes in section 2 which we will use as the basis for the notion of a hierarchical Dirichlet process (HDP) described in section 3. We explore properties of the HDP prior, showing that it can generate interesting hidden state sequences and that it can also be used as an emission model for an infinite alphabet of symbols. This infinite emission model is controlled by two additional hyperparameters. In section 4 we describe the procedures for inference (Gibbs sampling the hidden states), learning (optimising the hyperparameters), and likelihood evaluation (infinite-state particle filtering). We present experimental results in section 5 and conclude in section 6.

## 2  Properties of the Dirichlet Process

Let us examine in detail the statistics of hidden state transitions from a particular state $s_t = i$ to $s_{t+1}$, with the number of hidden states **finite** and equal to $k$. The transition probabilities given in the $i^{\text{th}}$ row of the transition matrix can be interpreted as mixing proportions for $s_{t+1}$ that we call $\boldsymbol{\pi} = \{\pi_1, \ldots, \pi_k\}$.

Imagine drawing $n$ samples $\{c_1, \ldots, c_n\}$ from a discrete indicator variable which can take on values $\{1, \ldots, k\}$ with proportions given by $\boldsymbol{\pi}$. The joint distribution of these indicators is multinomial

$$P(c_1, \ldots, c_n | \boldsymbol{\pi}) = \prod_{j=1}^{k} \pi_j^{n_j} , \qquad \text{with} \quad n_j = \sum_{n'=1}^{n} \delta(c_{n'}, j) \tag{1}$$

where we have used the Kronecker-delta function ($\delta(a, b) = 1$ iff $a = b$, and 0 otherwise) to count the number of times $n_j$ that $s_{t+1} = j$ has been drawn. Let us see what happens to the distribution of these indicators when we integrate out the mixing proportions $\boldsymbol{\pi}$ under a conjugate prior. We give the mixing proportions a symmetric Dirichlet prior with positive *concentration* hyperparameter $\beta$

$$P(\boldsymbol{\pi} | \beta) \sim \text{Dirichlet}(\beta/k, \ldots, \beta/k) \; = \; \frac{\Gamma(\beta)}{\Gamma(\beta/k)^k} \prod_{j=1}^{k} \pi_j^{\beta/k-1} , \tag{2}$$

where $\boldsymbol{\pi}$ is restricted to be on the simplex of mixing proportions that sum to 1. We can analytically integrate out $\boldsymbol{\pi}$ under this prior to yield:

$$P(c_1, \ldots, c_n | \beta) = \int d\boldsymbol{\pi} \ P(c_1, \ldots, c_n | \boldsymbol{\pi}) P(\boldsymbol{\pi} | \beta) = \frac{\Gamma(\beta)}{\Gamma(n + \beta)} \prod_{j=1}^{k} \frac{\Gamma(n_j + \beta/k)}{\Gamma(\beta/k)} \ . \quad (3)$$

Thus the probability of a particular sequence of indicators is only a function of the counts $\{n_1, \ldots, n_k\}$. The conditional probability of an indicator $c_d$ given the setting of all other indicators (denoted $\mathbf{c}_{-d}$) is given by

$$P(c_d = j | \mathbf{c}_{-d}, \beta) = \frac{n_{-d,j} + \beta/k}{n - 1 + \beta} \ , \quad (4)$$

where $n_{-d,j}$ is the counts as in (1) with the $d^{\text{th}}$ indicator removed. Note the self-reinforcing property of (4): $c_d$ is more likely to choose an already popular state. A key property of DPs, which is at the very heart of the model in this paper, is the expression for (4) when we take the limit as the number of hidden states $k$ tends to infinity:

$$P(c_d = j | \mathbf{c}_{-d}, \beta) = \begin{cases} \frac{n_{-d,j}}{n-1+\beta} & j \in \{1, \ldots, K\} \text{ i.e. represented} \\ \frac{\beta}{n-1+\beta} & \text{for all unrepresented } j, \text{ combined} \end{cases} \quad (5)$$

where $K$ is the number of represented states (i.e. for which $n_{-d,j} > 0$), which cannot be infinite since $n$ is finite. $\beta$ can be interpreted as the number of pseudo-observations of $\boldsymbol{\pi} = \{1/k, \ldots, 1/k\}$, i.e. the *strength* of belief in the symmetric prior.[2] In the infinite limit $\beta$ acts as an "innovation" parameter, controlling the tendency for the model to populate a previously unrepresented state.

## 3 Hierarchical Dirichlet Process (HDP)

We now consider modelling each row of the transition and emission matrices of an HMM as a DP. Two key results from the previous section form the basis of the HDP model for infinite HMMs. The first is that we can integrate out the infinite number of transition parameters, and represent the process with a finite number of indicator variables. The second is that under a DP there is a natural tendency to use existing transitions in proportion to their previous usage, which gives rise to *typical trajectories*. In sections 3.1 and 3.2 we describe in detail the HDP model for transitions and emissions for an infinite-state HMM.

### 3.1 Hidden state transition mechanism

Imagine we have generated a hidden state sequence up to and including time $t$, building a table of counts $n_{ij}$ for transitions that have occured so far from state $i$ to $j$, i.e. $n_{ij} = \sum_{t'=1}^{t-1} \delta(s_{t'}, i)\delta(s_{t'+1}, j)$. Given that we are in state $s_t = i$, we impose on state $s_{t+1}$ a DP (5) with parameter $\beta$ whose counts are those entries in the $i^{\text{th}}$ row of $n$, i.e. we prefer to reuse transitions we have used before and follow typical trajectories (see Figure 1):

$$P(s_{t+1} = j | s_t = i, n, \beta) = \frac{n_{ij}}{\sum_{j'=1}^{K} n_{ij'} + \beta} \quad j \in \{1, \ldots, K\} \ . \quad (6)$$

Note that the above probabilities do not sum to 1—under the DP there is a finite probability $\beta/(\sum_{j'} n_{ij'} + \beta)$ of not selecting one of these transitions. In this case, the model defaults to a second different DP (5) on $s_{t+1}$ with parameter $\gamma$ whose counts are given by a vector $n_j^o$. We refer to the default DP and its associated counts as the *oracle*. Given that we have defaulted to the oracle DP, the probabilities of transitioning now become

$$P(s_{t+1} = j | s_t = i, n^o, \gamma) = \begin{cases} \frac{n_j^o}{\sum_{j'=1}^{K} n_{j'}^o + \gamma} & j \in \{1, \ldots, K\} \text{ i.e. } j \text{ represented} , \\ \frac{\gamma}{\sum_{j'=1}^{K} n_{j'}^o + \gamma} & j \notin \{1, \ldots, K\} \text{ i.e. } j \text{ is a \textbf{new} state} . \end{cases} \quad (7)$$

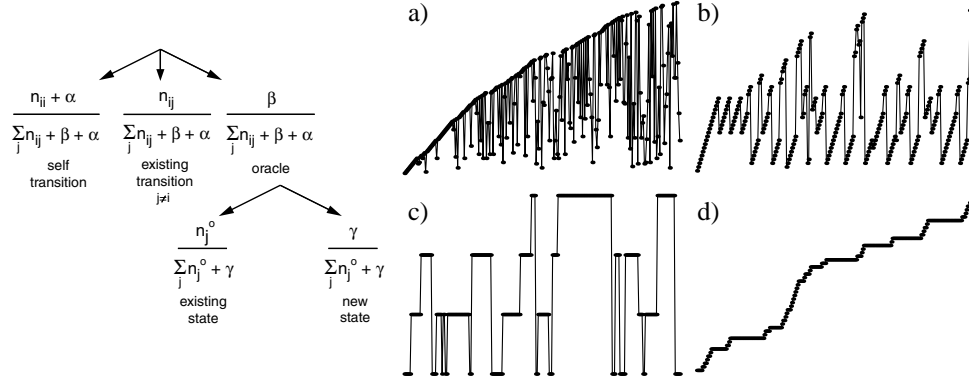

Figure 1: **(left)** State transition generative mechanism. **(right a-d)** Sampled state trajectories of length $T = 250$ (time along horizontal axis) from the HDP: we give examples of four modes of behaviour. **(a)** $\alpha = 0.1, \beta = 1000, \gamma = 100$, explores many states with a sparse transition matrix. **(b)** $\alpha = 0, \beta = 0.1, \gamma = 100$, retraces multiple interacting trajectory segments. **(c)** $\alpha = 8, \beta = 2, \gamma = 2$, switches between a few different states. **(d)** $\alpha = 1, \beta = 1, \gamma = 10000$, has strict left-to-right transition dynamics with long linger time.

Under the oracle, with probability proportional to $\gamma$ an entirely new state is transitioned to. This is the only mechanism for visiting new states from the infinitely many available to us. After each transition we set $n_{ij} \leftarrow n_{ij} + 1$ and, if we transitioned to the state $j$ via the oracle DP just described then in addition we set $n_j^o \leftarrow n_j^o + 1$. If we transitioned to a new state then the size of $n$ and $n^o$ will increase.

Self-transitions are special because their probability defines a time scale over which the dynamics of the hidden state evolves. We assign a finite prior mass $\alpha$ to self transitions for each state; this is the third hyperparameter in our model. Therefore, when first visited (via $\gamma$ in the HDP), its self-transition count is initialised to $\alpha$.

The full hidden state transition mechanism is a two-level DP hierarchy shown in decision tree form in Figure 1. Alongside are shown typical state trajectories under the prior with different hyperparameters. We can see that, with just three hyperparameters, there are a wealth of types of possible trajectories. Note that $\gamma$ controls the expected number of represented hidden states, and $\beta$ influences the tendency to explore new transitions, corresponding to the *size* and *density* respectively of the resulting transition count matrix. Finally $\alpha$ controls the prior tendency to linger in a state.

The role of the oracle is two-fold. First it serves to couple the transition DPs from different hidden states. Since a newly visited state has no previous transitions to existing states, without an oracle (which necessarily has knowledge of all represented states as it created them) it would transition to itself or yet another new state with probability 1. By consulting the oracle, new states can have finite probability of transitioning to represented states. The second role of the oracle is to allow some states to be more influential (more commonly transitioned to) than others.

## 3.2 Emission mechanism

The emission process $s_t \rightarrow y_t$ is identical to the transition process $s_t \rightarrow s_{t+1}$ in every respect except that there is no concept analogous to a self-transition. Therefore we need only introduce two further hyperparameters $\beta^e$ and $\gamma^e$ for the emission HDP. Like for state transitions we keep a table of counts $m_{iq} = \sum_{t'=1}^{t-1} \delta(s_{t'}, i)\delta(y_{t'}, q)$ which is the number of times before $t$ that state $i$ has emitted symbol $q$, and $m_q^o$ is the number of times symbol

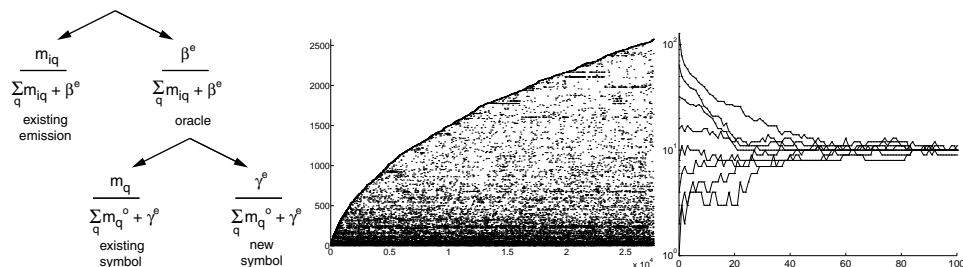

Figure 2:   **(left)** State emission generative mechanism. **(middle)** Word occurence for entire *Alice* novel: each word is assigned a unique integer identity as it appears. Word identity (vertical) is plotted against the word position (horizontal) in the text. **(right)** (Exp 1) Evolution of number of represented states $K$ (vertical), plotted against iterations of Gibbs sweeps (horizontal) during learning of the *ascending-descending* sequence which requires exactly 10 states to model the data perfectly. Each line represents initialising the hidden state to a random sequence containing $K = \{1, 2, 4, \ldots, 128\}$ distinct represented states. (Hyperparameters are not optimised.)

$q$ has been emitted using the emission oracle.

For some applications the training sequence is not expected to contain all possible observation symbols. Consider the occurence of words in natural text e.g. as shown in Figure 2 (middle) for the *Alice* novel. The upper envelope demonstrates that new words continue to appear in the novel. A property of the DP is that the expected number of distinct symbols (i.e. words here) increases as the logarithm of the sequence length. The combination of an HDP for both hidden states and emissions may well be able to capture the somewhat super-logarithmic word generation found in *Alice*.

# 4   Inference, learning and likelihoods

Given a sequence of observations, there are two sets of unknowns in the infinite HMM: the hidden state sequence $\mathbf{s} = \{s_1, \ldots, s_T\}$, and the five hyperparameters $\{\alpha, \beta, \gamma, \beta^e, \gamma^e\}$ defining the transition and emission HDPs. Note that by using HDPs for both states and observations, we have implicitly integrated out the infinitely many transition and emission parameters. Making an analogy with non-parametric models such as Gaussian Processes, we define a learned model as a set of counts $\{n, n^o, m, m^o\}$ and optimised hyperparameters $\{\alpha, \beta, \gamma, \beta^e, \gamma^e\}$.

We first describe an approximate Gibbs sampling procedure for inferring the posterior over the hidden state sequence. We then describe hyperparameter optimisation. Lastly, for calculating the likelihood we introduce an infinite-state particle filter. The following algorithm summarises the learning procedure:

1. Instantiate a random hidden state sequence $\{s_1, \ldots, s_T\}$.
2. For $t = 1, \ldots, T$
   - Gibbs sample $s_t$ given hyperparameter settings, count matrices, and observations.
   - Update count matrices to reflect new $s_t$; this may change $K$, the number of represented hidden states.
3. End $t$
4. Update hyperparameters $\{\alpha, \beta, \gamma, \beta^e, \gamma^e\}$ given hidden state statistics.
5. Goto step 2.

## 4.1   Gibbs sampling the hidden state sequence

Define $\tilde{n}$ and $\tilde{m}$ as the results of removing from $n$ and $m$ the transition and emission counts contributed by $s_t$. Define similar items $\tilde{n}^o$ and $\tilde{m}^o$ related to the transition and emission

oracle vectors. An exact Gibbs sweep of the hidden state from $t = 1, \ldots, T$ takes $\mathcal{O}(T^2)$ operations, since under the HDP generative process changing $s_t$ affects the probability of all subsequent hidden state transitions and emissions.[3] However this computation can be reasonably approximated in $\mathcal{O}(T)$, by basing the Gibbs update for $s_t$ only on the state of its neigbours $\{s_{t-1}, y_t, s_{t+1}\}$ and the total counts $\tilde{n}, \tilde{m}, \tilde{n}^o, \tilde{m}^o$.[4]

In order to facilitate hyperparameter learning and improve the mixing time of the Gibbs sampler, we also sample a set of auxiliary indicator variables $\{o_t, o_{t+1}, o_t^e\}$ alongside $s_t$; each of these is a binary variable denoting whether the oracle was used to generate $\{s_t, s_{t+1}, y_t\}$ respectively.

### 4.2 Hyperparameter optimisation

We place vague Gamma priors[5] on the hyperparameters $\{\alpha, \beta, \gamma, \beta^e, \gamma^e\}$. We derive an approximate form for the hyperparameter posteriors from (3) by treating each level of the HDPs separately. The following expressions for the posterior for $\alpha$, $\beta$, and $\beta^e$ are accurate for large $\gamma$, while the expressions for $\gamma$ and $\gamma^e$ are exact:

$$P(\alpha, \beta | \mathbf{s}) \propto \mathcal{G}(a_\alpha, b_\alpha) \mathcal{G}(a_\beta, b_\beta) \prod_{i=1}^{K} \frac{\beta^{K^{(i)}-1} \Gamma(\alpha + \beta)}{\Gamma(\alpha)} \frac{\Gamma(n_{ii} + \alpha)}{\Gamma(\sum_j n_{ij} + \alpha + \beta)} ,$$

$$P(\beta^e | \mathbf{s}, \mathbf{y}) \propto \mathcal{G}(a_{\beta^e}, b_{\beta^e}) \prod_{i=1}^{K} \frac{\beta^{e K^{e(i)}} \Gamma(\beta^e)}{\Gamma(\sum_q m_{iq} + \beta^e)} ,$$

$$P(\gamma | \mathbf{s}) \propto \mathcal{G}(a_\gamma, b_\gamma) \frac{\gamma^K \Gamma(\gamma)}{\Gamma(T^o + \gamma)} , \qquad P(\gamma^e | \mathbf{s}, \mathbf{y}) \propto \mathcal{G}(a_{\gamma^e}, b_{\gamma^e}) \frac{\gamma^{K^e} \Gamma(\gamma^e)}{\Gamma(T^{oe} + \gamma^e)}$$

where $K^{(i)}$ is the number of represented states that are transitioned to from state $i$ (including itself); similarly $K^{e(i)}$ is the number of possible emissions from state $i$. $T^o$ and $T^{oe}$ are the number of times the oracle has been used for the transition and emission processes, calculated from the indicator variables $\{o_t, o_t^e\}$. We solve for the maximum a posteriori (MAP) setting for each hyperparameter; for example $\beta_{\text{MAP}}^e$ is obtained as the solution to following equation using gradient following techniques such as Newton-Raphson:

$$\sum_{i=1}^{K} \left[ K^{e(i)} / \beta_{\text{MAP}}^e + \psi(\beta_{\text{MAP}}^e) - \psi(\sum_q m_{iq} + \beta_{\text{MAP}}^e) \right] - b_{\beta^e} + (a_{\beta^e} - 1) / \beta_{\text{MAP}}^e = 0 .$$

### 4.3 Infinite-state particle filter

The likelihood for a particular observable sequence of symbols involves intractable sums over the possible hidden state trajectories. Integrating out the parameters in any HMM induces long range dependencies between states. In particular, in the DP, making the transition $i \rightarrow j$ makes that transition more likely later on in the sequence, so we cannot use standard tricks like dynamic programming. Furthermore, the number of distinct states can grow with the sequence length as new states are generated. If the chain starts with $K$ distinct states, at time $t$ there could be $K + t$ possible distinct states making the total number of trajectories over the entire length of the sequence $(K + T)! / K!$.

We propose estimating the likelihood of a test sequence given a learned model using particle filtering. The idea is to start with some number of particles $R$ distributed on the represented hidden states according to the final state marginal from the training sequence (some of the $R$ may fall onto new states).[6] Starting from the set of particles $\{s_t^1, \ldots, s_t^R\}$, the tables from the training sequences $\{n, n^o, m, m^o\}$, and $t = 1$ the recursive procedure is as specified below, where $P(s_t|y_1, \ldots, y_{t-1}) \approx \frac{1}{R} \sum_r \delta(s_t, s_t^r)$ :

1. Compute $l_t^r = P(y_t|s_t = s_t^r)$ for each particle $r$.
2. Calculate $l_t = (1/R) \sum_r l_t^r \approx P(y_t|y_1, \ldots, y_{t-1})$.
3. Resample $R$ particles $s_t^r \sim (1/\sum_{r'} l_t^{r'}) \sum_{r'} l_t^{r'} \delta(s_t, s_t^{r'})$.
4. Update transition and emission tables $n^r$, $m^r$ for each particle.
5. For each $r$ sample forward dynamics: $s_{t+1}^r \sim P(s_{t+1}|s_t = s_t^r, n^r, m^r)$; this may cause particles to land on novel states. Update $n^r$ and $m^r$.
6. If $t < T$, Goto 1 with $t = t + 1$.

The log likelihood of the test sequence is computed as $\sum_t \log l_t$. Since it is a discrete state space, with much of the probability mass concentrated on the represented states, it is feasible to use $O(K)$ particles.

## 5  Synthetic experiments

**Exp 1: Discovering the number of hidden states**   We applied the infinite HMM inference algorithm to the *ascending-descending* observation sequence consisting of 30 concatenated copies of $ABCDEFEDCB$. The most parsimonious HMM which models this data perfectly has exactly 10 hidden states. The infinite HMM was initialised with a random hidden state sequence, containing $K$ distinct represented states. In Figure 2 (right) we show how the number of represented states evolves with successive Gibbs sweeps, starting from a variety of initial $K$. In all cases $K$ converges to 10, while occasionally exploring 9 and 11.

**Exp 2: Expansive**   A sequence of length $T = 800$ was generated from a 4-state 8-symbol HMM with the transition and emission probabilities as shown in Figure 3 (top left).

**Exp 3: Compressive**   A sequence of length $T = 800$ was generated from a 4-state 3-symbol HMM with the transition and emission probabilities as shown in Figure 3 (bottom left).

In both Exp 2 and Exp 3 the infinite HMM was initialised with a hidden state sequence with $K = 20$ distinct states. Figure 3 shows that, over successive Gibbs sweeps and hyperparameter learning, the count matrices for the infinite HMM converge to resemble the true probability matrices as shown on the far left.

## 6  Discussion

We have shown how a two-level Hierarchical Dirichlet Process can be used to define a non-parametric Bayesian HMM. The HDP implicity integrates out the transition and emission parameters of the HMM. An advantage of this is that it is no longer necessary to constrain the HMM to have finitely many states and observation symbols. The prior over hidden state transitions defined by the HDP is capable of producing a wealth of interesting trajectories by varying the three hyperparameters that control it.

We have presented the necessary tools for using the infinite HMM, namely a linear-time approximate Gibbs sampler for inference, equations for hyperparameter learning, and a particle filter for likelihood evaluation.

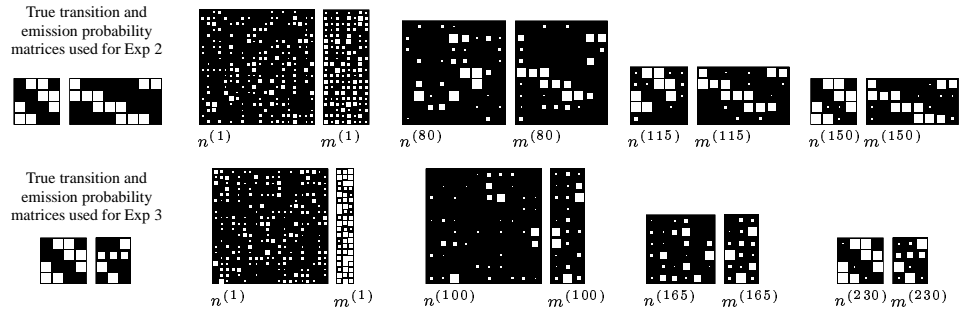

Figure 3: The far left pair of Hinton diagrams represent the true transition and emission probabilities used to generate the data for each experiment 2 and 3 (up to a permutation of the hidden states; lighter boxes correspond to higher values). **(top row)** Exp 2: Expansive HMM. Count matrix pairs $\{n, m\}$ are displayed after $\{1, 80, 115, 150\}$ sweeps of Gibbs sampling. **(bottom row)** Exp 3: Compressive HMM. Similar to top row displaying count matrices after $\{1, 100, 165, 230\}$ sweeps of Gibbs sampling. In both rows the display after a single Gibbs sweep has been reduced in size for clarity.

On synthetic data we have shown that the infinite HMM discovers both the appropriate number of states required to model the data and the structure of the emission and transition matrices. It is important to emphasise that although the count matrices found by the infinite HMM resemble point estimates of HMM parameters (e.g. Figure 3), they are better thought of as the sufficient statistics for the HDP posterior *distribution* over parameters.

We believe that for many problems the infinite HMM's flexibile nature and its ability to automatically determine the required number of hidden states make it superior to the conventional treatment of HMMs with its associated difficult model selection problem. While the results in this paper are promising, they are limited to synthetic data; in future we hope to explore the potential of this model on real-world problems.

### Acknowledgements

The authors would like to thank David Mackay for suggesting the use of an oracle, and Quaid Morris for his Perl expertise.

## Footnotes

[1] That is, if we only applied the mechanism described in section 2, then state trajectories under the prior would never visit the same state twice; since each new state will have no previous transitions from it, the DP would choose randomly between all infinitely many states, therefore transitioning to another new state with probability 1.

[2]Under the infinite model, at any time, there are an infinite number of (indistinguishable) unrepresented states available, each of which have infinitesimal mass proportional to $\beta$.

[3]Although the hidden states in an HMM satisfy the Markov condition, integrating out the parameters induces these long-range dependencies.

[4]This approximation can be motivated in the following way. Consider sampling parameters $\boldsymbol{\theta}$ from the posterior distribution $P(\boldsymbol{\theta} | \mathbf{y}, \mathbf{s})$ of parameter matrices, which will depend on the count matrices. By the Markov property, for a *given* $\boldsymbol{\theta}$, the probability of $s_t$ only depends on $s_{t-1}$, $y_t$ and $s_{t+1}$, and can therefore be computed without considering its effect on future states.

[5]$\nu \sim \mathcal{G}(a, b) = b^a / \Gamma(a) \cdot \nu^{a-1} e^{-b\nu}$, with $a$ and $b$ the shape and inverse-scale parameters.

[6]Different particle initialisations apply if we do not assume that the test sequence immediately follows the training sequence.

### References

[1] C. E. Antoniak. Mixtures of Dirichlet processes with applications to Bayesian nonparametric problems. *Annals of Statistics*, 2(6):1152–1174, 1974.

[2] T. S. Ferguson. A Bayesian analysis of some nonparametric problems. *Annals of Statistics*, 1(2):209–230, March 1973.

[3] D. J. C. MacKay. Ensemble learning for hidden Markov models. Technical report, Cavendish Laboratory, University of Cambridge, 1997.

[4] D. J. C. MacKay and L. C. Peto. A hierarchical Dirichlet language model. *Natural Language Engineering*, 1(3):1–19, 1995.

[5] R. M. Neal. Markov chain sampling methods for Dirichlet process mixture models. Technical Report 9815, Dept. of Statistics, University of Toronto, 1998.

[6] L. R. Rabiner and B. H. Juang. An introduction to hidden Markov models. *IEEE Acoustics, Speech & Signal Processing Magazine*, 3:4–16, 1986.

[7] C. E. Rasmussen. The infinite Gaussian mixture model. In *Advances in Neural Information Processing Systems 12*, Cambridge, MA, 2000. MIT Press.

[8] A. Stolcke and S. Omohundro. Hidden Markov model induction by Bayesian model merging. In S. J. Hanson, J. D. Cowan, and C. L. Giles, editors, *Advances in Neural Information Processing Systems 5*, pages 11–18, San Francisco, CA, 1993. Morgan Kaufmann.
